# Global Regularization of Inverse Kinematics for Redundant Manipulators

**David DeMers**
Dept. of Computer Science & Engr.
Institute for Neural Computation
University of California, San Diego
La Jolla, CA 92093-0114

**Kenneth Kreutz-Delgado**
Dept. of Electrical & Computer Engr.
Institute for Neural Computation
University of California, San Diego
La Jolla, CA 92093-0407

## Abstract

The inverse kinematics problem for redundant manipulators is ill–posed and nonlinear. There are two fundamentally different issues which result in the need for some form of regularization; the existence of multiple solution branches (global ill–posedness) and the existence of excess degrees of freedom (local ill–posedness). For certain classes of manipulators, learning methods applied to input–output data generated from the forward function can be used to globally regularize the problem by partitioning the domain of the forward mapping into a finite set of regions over which the inverse problem is well–posed. Local regularization can be accomplished by an appropriate parameterization of the redundancy consistently over each region. As a result, the ill–posed problem can be transformed into a finite set of well–posed problems. Each can then be solved separately to construct approximate direct inverse functions.

## 1 INTRODUCTION

The robot forward kinematics function maps a vector of joint variables to the end–effector configuration space, or *workspace*, here assumed to be Euclidean. We denote this mapping by $f(\cdot) : \Theta^n \to \mathcal{W}^m \subseteq \mathcal{X}^m$, $f(\theta) \mapsto \mathbf{x}$, for $\theta \in \Theta^n$ (the *input space* or *joint space*) and $\mathbf{x} \in \mathcal{W}^m$ (the *workspace*). When $m < n$, we say that the manipulator has redundant degrees–of–freedom (dof).

The inverse kinematics problem is the following: given a desired workspace location $\mathbf{x}$, find joint variables $\theta$ such that $f(\theta) = \mathbf{x}$. Even when the forward kinematics is known, the inverse kinematics for a manipulator is not generically solvable in closed form (Craig, 1986). This problem is ill-posed[1] due to two separate phenomena. First, multiple solution branches can exist (for both non–redundant as well as redundant manipulators). The second source of ill–posedness arises because of the redundant dofs. Each of the inverse solution branches consists of a submanifold of dimensionality equal to the number of redundant dofs. Thus the inverse solution requires two regularizations; *global* regularization to select a solution branch, and *local* regularization, to resolve the redundancy. In this paper the existence of at least one solution is assumed; that is, inverses will be sought only for points in the reachable workspace, i.e. desired x in the image of $f(\cdot)$.

Given input–output data generated from the kinematics mapping (pairs consisting of joint variable values & corresponding end–effector location), can the inverse mapping be learned without making any *a priori* regularizing assumptions or restrictions? We show that the answer can be "yes". The approach taken towards the solution is based on the use of learning methods to partition the data into groups such that the inverse kinematics problem, when restricted to each group, is well–posed, after which a direct inverse function can be approximated on each group.

A direct inverse function is desireable. For instance, a direct inverse is computable quickly; if implemented by a feedforward network were used, one function evaluation is equivalent to a single forward propagation. More importantly, theoretical results show that an algorithm for tracking a cyclic path in the workspace will produce a cyclic trajectory of joint angles if and only if it is equivalent to a direct inverse function (Baker, 1990). That is, inverse functions are necessary to ensure that when following a closed loop the arm configurations which result in the same end–effector location will be the same.

Unfortunately, topological results show that a single *global* inverse function does not exist for generic robot manipulators. However, a global topological analysis of the kinematics function and the nature of the manifolds induced in the input space and workspace show that for certain robot geometries the mapping may be expressed as the union of a finite set of well–behaved local regions (Burdick, 1991). In this case, the redundancy takes the form of a submanifold which can be parameterized (locally) consistently by, for example, the use of topology preserving neural networks.

## 2   TOPOLOGY AND ROBOT KINEMATICS

It is known that for certain robot geometries the input space can be partitioned into disjoint regions which have the property that no more than one inverse solution branch lies within any one of the regions (Burdick, 1988). We assume in the following that the manipulator in question has such a geometry, and has all revolute joints. Thus $\Theta^n = T^n$, the $n$–torus. The redundancy manifolds in this case have the topology of $T^{n-m}$, $n - m$–dimensional torii.

For $\Theta^n$ a compact manifold of dimensionality $n$, $\mathcal{W}^m$ a compact manifold of dimensionality $m$, and $f$ a smooth map from $\Theta^n$ to $\mathcal{W}^m$, let the differential $d_\Theta f$ be the map from the tangent space of $\Theta^n$ at $\theta \in \Theta^n$ to the tangent space of $\mathcal{W}^m$ at $f(\theta)$. The set of points in $\Theta^n$ which

map to $x \in \mathcal{W}^m$ is the *pre–image* of $x$, denoted by $f^{-1}(x)$. The differential $d_\Theta f$ has a natural representation given by an $m$ by $n$ *Jacobian* matrix whose elements consist of the first partial derivatives of $f$ w.r.t. a basis of $\Theta^n$. Define $\mathcal{S}$ as the set of *critical points* of $f$, which are the set of all $\theta \in \Theta^n$ such that $d_\Theta f(\theta)$ has rank less than the dimensionality of $\mathcal{W}^m$. Elements of the image of $\mathcal{S}$, $f(\mathcal{S})$ are called the *critical values*. The set $\mathcal{R} \overset{\Delta}{=} \mathcal{W}^m \backslash \mathcal{S}$ are the *regular values* of $f$. For $\theta \in \Theta^n$, if $\exists \theta^* \in f^{-1}(f(\theta))$, $\theta^* \in \mathcal{S}$, we call $\theta$ a *co–regular point* of $f$.

The kinematic mapping of certain classes of manipulators (with the geometry herein assumed) can be decomposed based on the co–regular surfaces which divide $\Theta^n$ into a finite number of disjoint, connected regions, $C_i$. The image of each $C_i$ under $f$ is a connected region in the workspace, $\mathcal{W}_i$. We denote the kinematics mapping restricted to $C_i$ to be $f_i$; $f_i : C_i \to \mathcal{W}_i$.

Locally, one inverse solution branch for a region in the workspace has the structure of a product space. We conjecture that $(\mathcal{W}_i, T^{n-m}, C_i, f|_{C_i})$ forms a locally trivial fiber bundle[2] and that the $C_i$, therefore, form regions where the inverse is unique modulo the redundancy.

Given a point in the workspace, $x$, for which a configuration is sought, global regularization requires choosing from among the multiple pre–image torii. Local regularization (redundancy resolution) requires finding a location on the chosen torus. We would like to effectively "mod out" the redundancy manifolds by constructing an indexed one–to–one, invertible mapping from each pre–image manifold to a point in $\mathcal{X}^m$, and to obtain a consistent representation of the manifold by constructing an invertible mapping from itself to a set of $n - m$ "location" parameters.

# 3  GLOBAL REGULARIZATION

The existence of multiple solutions for even non–redundant manipulators poses difficult problems. Usually (and often for plausible reasons) the manipulator's allowable configurations or task space is effectively constrained so that there exists only a single inverse solution (Martinetz et al., 1990; Kuperstein, 1991). This approach regularizes the problem by allowing the existence of only one–to–one data. We seek to generalize to the multi–solution case and to learn all of the possible solutions.

For an all revolute manipulator there typically will be multiple pre–image torii for a particular point in the workspace. The topology of the pre–image solution branches will generally be known from the type of geometry, although their number may not be obvious by inspection, and will usually be different for different regions of the workspace. An upper bound on the number of inverse solutions is known (Burdick, 1988); consequently, the determination of the number could be made by search for the best fit among the possibilities.

The sampling and clustering approach described in (DeMers & Kreutz–Delgado, 1992) can be used to partition input–output data into disjoint pre–image sets. This approach uses samples of the forward behavior to identify the sets in the input space which map to

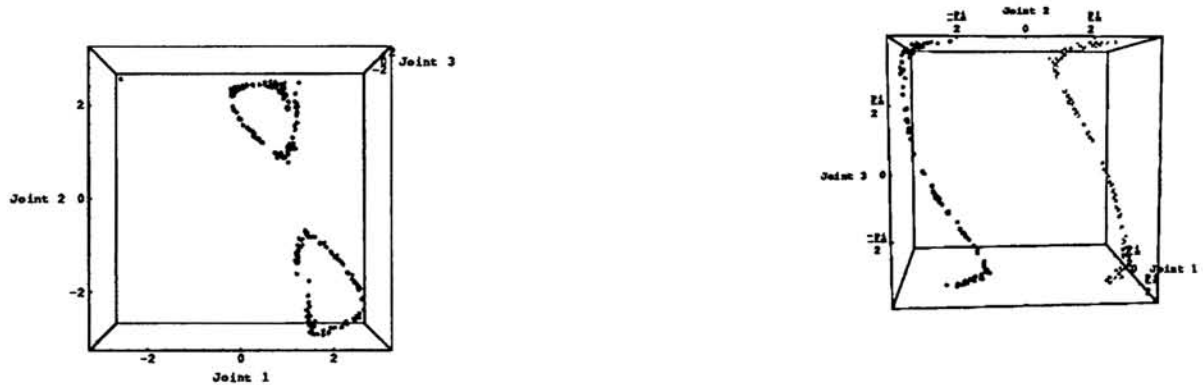

Figure 1: *The two pre–image manifolds for positioning the end–effector of the 3–R planar manipulator at a specific (x, y) location.*

within a small distance of a specific location in the workspace. The pre–image points will lie on the disjoint pre–image manifolds. These manifolds will typically be separable by clustering[3]. Figures 1 shows two views of the two redundancy, or "self–motion" manifolds for a particular end–effector location of the 3–R planar arm. All of the input space points shown position the arm's end–effector near the same (x,y) location. Note that the input space is the 3–torus, $T^3$. In order to visualize the space, the torus is "sliced" along each dimension. Thus opposite faces of the "cube" shown are identified with each other.

## 4  LOCAL REGULARIZATION

The inverse kinematics problem for manipulators with redundant dofs is usually solved either by using differential methods, which attempt to exploit the redundancy by optimizing a task–dependent objective function, or by using learning methods which regularize at training time by adding constraints equal to the number of redundant dof. The former may be computationally inefficient in that iterative solution of derivatives or matrix inversions are required, and may be unsatisfactory for real–time control. The latter is unsatisfying as it eliminates the run–time dexterity available from the redundancy; that is, it imposes prior constraints on the use of any extra dofs.

Although in practice numerical, differential methods are used for redundancy resolution, it has recently been shown that simple recurrent neural networks can resolve the redundancy by optimization of certain side–constraints at run–time, (Jordan & Rumelhart, 1992), (Kindermann & Linden, 1990). The differential methods have a number of desireable properties. In general it is possible to iterate in order to achieve a solution of arbitrary accuracy. They also tend to be capable of handling very flexible constraints. Global regularization as discussed above can be used to augment such methods. For example, once a choice of a solution branch has been made, an initial starting location away from singularities can be selected, and differential methods used to achieve an accurate solution on that branch.

Our work shows that construction of redundancy–parameterized approximations to di-

rect inverses are achievable. That is, the mapping from the workspace (augmented by a parameterization of the redundancy) to the input space can be approximated. This local regularization is accomplished by parameterizing the pre–image solution branch torii. Given (enough) samples of $\theta$ points paired with their $x$ image, a parameterization can be discovered for each branch. The method used exploits the fact that neighborhoods within each $C_i$ map to neighborhoods, and that neighborhoods within each $W_i$ have as pre-images a finite number of solution branches.

First, all points in our sample which have their image near some initial point $x_0$ are found. Pulling back to the input space by accessing the $\theta$ component of each of these $(\theta, x)$ data pairs finds the points in the pre–image set of the neighborhood of $x_0$. Now, because the topology of the pre–image set is known (here, the torus), a self–organizing map of appropriate topology can be fit to the pre–image points in order to parameterize this manifold. Neighboring torii have similar parameterizations; thus by repeating this process for a point $x_1$ near $x_0$ and using the parameterization of the pre–image of $x_0$ as initial conditions, a parameterization of the pre–image of $x_1$ qualitatively similar to that for $x_0$ can be constructed efficiently. By stepping between such "query points", $x_i$ a set of parameterizations can be obtained.

## 5   THE 3–R REDUNDANT PLANAR ARM

This approach can be used to provide a global and local regularization for a three–link manipulator performing the task of positioning in the plane. For this manipulator, $f : T^3 \rightarrow R^2$. The map $f$ restricted to each connected region in the input space bounded by the co–regular separating surfaces defines $f_i : T \times R \times S^1 \rightarrow R \times S^1$.

The pre–image of a point in the workspace thus consists of either one or two 1–torii (the actual number can be no more than the number of inverse solutions for a non–redundant manipulator of the same type, (Burdick, 1988)). Each torus is the pre–image of one of the restricted mappings $f_i$. The goal is to identify these torii and parameterize them. Figure 2 shows the input space and workspace of this arm, and their separating surfaces. These partition the workspace into disjoint annular regions, and the input space into disjoint tubular regions. The circles indicate workspace locations which can be reached in a kinematically singular configuration. The inverse image of these circles form the co–regular separating surfaces in the input space. For the link lengths used here ($l_1 = 5$, $l_2 = 4$, $l_3 = 3$), there is a single self–motion manifold for workspace locations in regions A and C, and two self–motion manifolds for workspace locations in B and D. Figure 3 shows two views of the data points near the pre–image manifolds for an end effector location in region A, and its parameterization by a self–organizing map (using the elastic net algorithm). Such a parameterization can be made for various locations in the workspace. Inverse kinematics can thus be computed by first locating the nearest parameterization network for a given workspace position and then choosing a configuration on the manifold, which can be done at run–time. Because the kinematics map is locally smooth, interpolation between networks and nodes on the networks can be done for greater accuracy. For convenience, a node on each torus was chosen as a canonical zero-point, and the remaining nodes assigned values based on their normalized distance from this point. Therefore all parameter values are scaled to be in the interval $[0,1]$.

For some end–effector locations, there are two pre–image manifolds. These first need to be identified by a global partitioning, then the individual manifolds parameterized. The

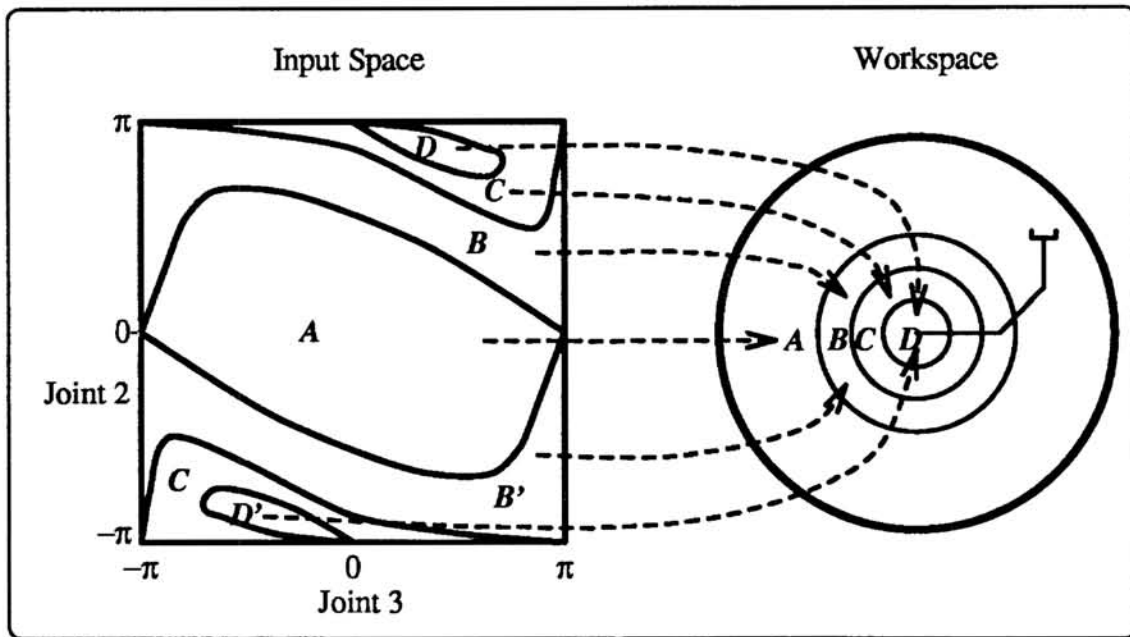

Figure 2: *The forward kinematics for a generic three–link planar manipulator. The separating surfaces in the input space do not depend on the value of joint angle 1, therefore the input space is shown projected to the joint 2 - joint 3 space. All end–effector locations inside regions A and C of the workspace have a single pre–image manifold in A and C, respectively, of the input space. End–effector locations inside regions B and D of the workspace have two pre–image manifolds, one in each of B and B' (resp. D and D') of the input space.*

manifolds may belong to one of a finite set of homotopy classes; that is, because they "live" in an ambient space which is a torus, they may or may not wrap around one or more of the dimensions of the torus. Unlike in Euclidean space, where there are only two possible one–dimensional manifolds, there are multiple topologically distinct types (homotopy classes) of closed loops which can serve as self–motion manifolds. Fortunately, because physical robots rarely have joints with unlimited range of motion, in practice the manifolds will usually not have wraparound. However, we should like to be able to parameterize any possibility. Appropriate choice of topology for a topology–preserving net results in an effective parameterization. Figure 4 shows two views of a parameterization for one of the self–motion manifolds shown above in Figure 1, which is the pre–image for an end–effector location in region B of Figure 2.

# 6   DISCUSSION

The global regularization accomplished by the method described above partitions the original input/output data into sets for each of the distinct $C_i$ regions. The redundancy parameters, $t$, obtained by local regularization can be used to augment this data, resulting in a transformation of the $(\theta, \mathbf{x})$ data into $(\theta_i, (\mathbf{x}_i, t))$. Let $\mathcal{T} : \theta \mapsto t$ be a function that computes a parameter value for each $\theta$ in the input space. Let $\hat{f}_i(\theta) = (f_i(\theta), \mathcal{T}(\theta))$. By construction, the regularized mapping $\hat{f}_i : \theta \mapsto (\mathbf{x}, t)$ is one–to–one and onto. Now, given examples from a one–to–one mapping, the inverse map $\hat{f}_i^{-1}(\mathbf{x}, t) \mapsto \theta$ can be directly approximated by, e.g., a feedforward neural network.

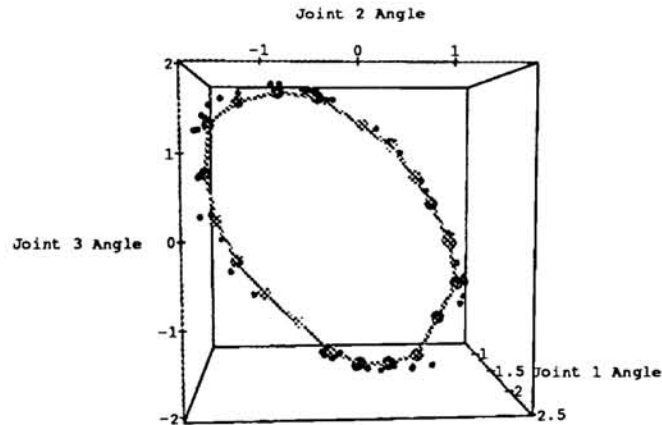

**Figure 3:** *The data points in the "self–motion" pre–image manifold of a point in the workspace of the 3–R planar arm, and a closed, 1–D elastic network after adaptation to them. This manifold is smoothly contractible to a point since it does not "wrap around" any of the dimensions of $T^3$.*

This method requires data sample sizes exponential in the number of degrees of freedom of the manipulator and thus will not be adequate for large dof "snake" manipulators. However, practical industrial robots of 7–dof may be amenable to our technique, especially if, as is common, it is designed with a separable wrist and is thus composable into a 4–dof redundant positioner plus a 3–dof non–redundant orienter.

This work can also be used to augment the differential methods of redundancy resolution. An approximate solution can be found extremely rapidly, and used to initialize gradient-based methods, which can then iterate to achieve a highly accurate solution. Global decisions such as choosing between multiple manifolds and identifying criteria for choosing locations on the manifold can now be made at run–time. Computation of an approximate direct inverse can then be made in constant time.

### Acknowledgements

This work was supported in part by NSF Presidential Young Investigator award IRI–9057631 and Fellowships from the California Space Institute and the McDonnell-Pew Center for Cognitive Neuroscience. The first author would like to thank the NIPS Foundation for providing student travel grants.

## Footnotes

[1]Ill–posedness can arise from having either too many or too few constraints to result in a unique and valid solution. That is, an overconstrained system may be ill–posed and have no solutions; such systems are typically solved by finding a least–squares or some such minimum cost solution. An underconstrained system may have multiple (possibly infinite) solutions. The inverse kinematics problem for redundant manipulators is underconstrained.

[2]A fiber bundle is a four-tuple consisting of a **base space**, a **fiber** (here, $T^{m-n}$), a **total space** and a projection $p$ mapping the total space to the base space with certain properties (here, the projection is equivalent to $f_i$ restricted to the total space). A locally trivial fiber bundle is one for which a consistent parameterization of the fibers is possible.

[3]For end–effector locations near the co–regular values, the pre–image manifolds tend to merge. This phenomenon can be identified by our methods.

### References

Daniel Baker (1990), "Some Topological Problems in Robotics", *The Mathematical Intelligencer*, Vol. 12, No. 1, pp. 66–76.

Joel Burdick (1988), "Kinematics and Design of Redundant Robot Manipulators", Stanford Ph.D. Thesis, Dept. of Mechanical Engineering.

Joel Burdick (1991), "A Classification of 3R Regional Manipulator Singularities and Geometries", *Proc. 1991 IEEE Intl. Conf. Robotics & Automation*, Sacramento.

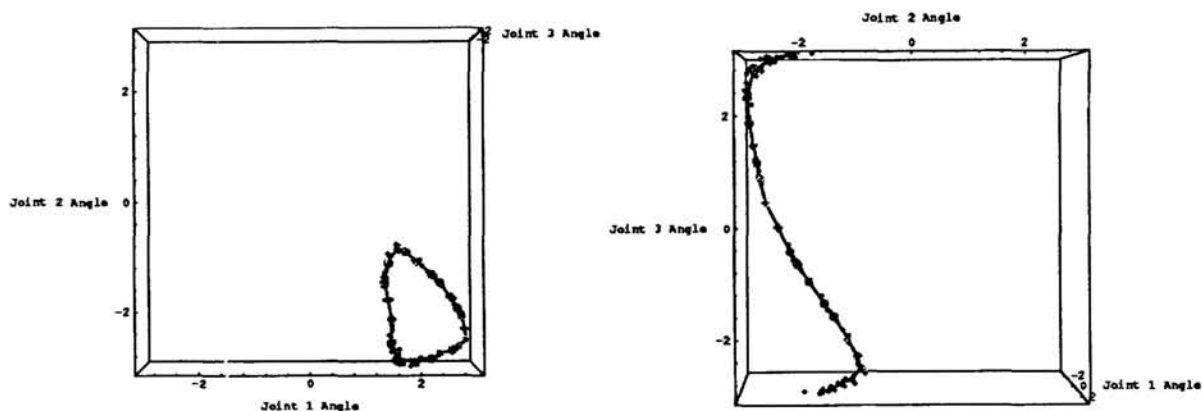

Figure 4: *Two views of the same data points in one of the two "self–motion" pre–image manifolds of a point in region B, and an elastic net after adaptation. It belongs to a different homotopy class than that of Fig 3 – it is not contractible to a point.*

John Craig (1986), *Introduction to Robotics.*

David DeMers & Kenneth Kreutz-Delgado (1992), "Learning Global Direct Inverse Kinematics", in Moody, J. E., Hanson, S.J. and Lippmann, R.P., eds, *Advances in Neural Information Processing Systems 4*, 589–594.

Michael I. Jordan & David Rumelhart (1992), "Forward Models: Supervised Learning with a Distal Teacher", *Cognitive Science 16*, 307–354.

J. Kindermann & Alexander Linden (1990), "Inversion of Neural Networks by Gradient Descent", *J. Parallel Computing 14*, 277–286.

Michael Kuperstein (1991), "INFANT Neural Controller for Adaptive Sensory–Motor Control", *Neural Networks*, Vol. 4, pp. 131–145.

Thomas Martinetz, Helge Ritter, & Klaus Schulten (1990), "Three–Dimensional Neural Networks for Learning Visuomotor Coordination of a Robot Arm", *IEEE Trans. Neural Networks*, Vol. 1, No. 1.

Charles Nash & Siddhartha Sen (1983), *Topology and Geometry for Physicists.*

Philippe Wenger (1992), "On the Kinematics of Manipulators with General Geometry: Application to the Feasibility Analysis of Continuous Trajectories", in M. Jamshidi, et al., eds, *Robotics and Manufacturing: Recent Trends in Research, Education and Applications 4*, 15–20 (ISRAM–92, Santa Fe).